# Decoding Cursive Scripts

**Yoram Singer**  and  **Naftali Tishby**
Institute of Computer Science and
Center for Neural Computation
Hebrew University, Jerusalem 91904, Israel

## Abstract

Online cursive handwriting recognition is currently one of the most intriguing challenges in pattern recognition. This study presents a novel approach to this problem which is composed of two complementary phases. The first is dynamic encoding of the writing trajectory into a compact sequence of discrete motor control symbols. In this compact representation we largely remove the redundancy of the script, while preserving most of its intelligible components. In the second phase these control sequences are used to train adaptive probabilistic acyclic automata (PAA) for the important ingredients of the writing trajectories, e.g. letters. We present a new and efficient learning algorithm for such stochastic automata, and demonstrate its utility for spotting and segmentation of cursive scripts. Our experiments show that over 90% of the letters are correctly spotted and identified, *prior* to any higher level language model. Moreover, both the training and recognition algorithms are very efficient compared to other modeling methods, and the models are 'on-line' adaptable to other writers and styles.

## 1   Introduction

While the emerging technology of pen-computing is already available on the world's markets, there is an on growing gap between the state of the hardware and the quality of the available online handwriting recognition algorithms. Clearly, the critical requirement for the success of this technology is the availability of reliable and robust cursive handwriting recognition methods.

We have previously proposed a dynamic encoding scheme for cursive handwriting based on an oscillatory model of handwriting [8, 9] and demonstrated its power mainly through analysis by synthesis. Here we continue with this paradigm and use the dynamic encoding scheme as the front-end for a complete stochastic model of cursive script.

The accumulated experience in temporal pattern recognition in the past 30 years has yielded some important lessons relevant to handwriting. The first is that one can not predefine the basic 'units' of such temporal patterns due to the strong inter-action, or 'coarticulation', between such units. Any reasonable model must allow for the large variability of the basic handwriting components in different contexts and by different writers. Thus true adaptability is a key ingredient of a good stochas-tic model of handwriting. Most, if not all, currently used models of handwriting and speech are hard to adapt and require vast amounts of training data for some robustness in performance. In this paper we propose a simpler stochastic modeling scheme, which we call Probabilistic Acyclic Automata (PAA), with the important feature of being adaptive. The training algorithm modifies the architecture and dimensionality of the model while optimizing its predictive power. This is achieved through the minimization of the "description length" of the model and training sequences, following the minimum description length (MDL) principle. Another interesting feature of our algorithm is that precisely the same procedure is used in both training and recognition phases, which enables continuous adaptation.

The structure of the paper is as follows. In section 2 we review our dynamic en-coding method, used as the front-end to the stochastic modeling phase. We briefly describe the estimation and quantization process, and show how the discrete motor control sequences are estimated and used, in section 3. Section 4 deals with our stochastic modeling approach and the PAA learning algorithm. The algorithm is demonstrated by the modeling of handwritten letters. Sections 5 and 6 deal with preliminary applications of our approach to segmentation and recognition of cursive handwriting.

## 2   Dynamic encoding of cursive handwriting

Motivated by the oscillatory motion model of handwriting, as described e.g. by Hollerbach in 1981 [2], we developed a parameter estimation and regularization method which serves for the analysis, synthesis and coding of cursive handwriting. This regularization technique results in a compact and efficient discrete representa-tion of handwriting.

Handwriting is generated by the human muscular motor system, which can be sim-plified as spring muscles near a mechanical equilibrium state. When the movements are small it is justified to assume that the spring muscles operate in the linear regime, so the basic movements are simple harmonic oscillations, superimposed by a simple linear drift. Movements are excited by selecting a pair of agonist-antagonist muscles that are modeled by the spring pair. In a restricted form this simple motion is described by the following two equations,

$$V_x(t) = \dot{x}(t) = a\cos(\omega_x t + \phi) + c \quad V_y(t) = \dot{y}(t) = b\cos(\omega_y t) \quad, \qquad (1)$$

where $V_x(t)$ and $V_y(t)$ are the horizontal and vertical pen velocities respectively, $\omega_x$ and $\omega_y$ are the angular velocities, $a, b$ are the velocity amplitudes, $\phi$ is the relative

phase lag, and $c$ is the horizontal drift velocity. Assuming that these describe the true trajectory, the horizontal drift, $c$, is estimated as the average horizontal velocity, $\dot{c} = \frac{1}{N} \sum_{i=1}^{N} V_x(i)$. For fixed values of the parameters $a, b, \omega$ and $\phi$ these equations describe a cycloidal trajectory.

Our main assumption is that the cycloidal trajectory is the natural (free) pen motion, which is modified only at the velocity zero crossings. Thus changes in the dynamical parameters occur only at the zero crossings and preserve the continuity of the velocity field. This assumption implies that the angular velocities $\omega_x, \omega_y$ and amplitudes $a, b$ can be considered constant between consecutive zero crossings. Denoting by $t_i^x$ and $t_i^y$, the i'th zero crossing locations of the horizontal and vertical velocities, and by $L_i^x$ and $L_i^y$, the horizontal and vertical progression during the i'th interval, then the estimated amplitudes are, $a = \frac{L_i^x \pi}{2(t_{i+1}^x - t_i^x)}$ , $b = \frac{L_i^y \pi}{2(t_{i+1}^y - t_i^y)}$. Those amplitudes define the vertical and horizontal scales of the written letters.

Examination of the vertical velocity dynamics reveals the following: **(a)** There is a virtual center of the vertical movement and velocity trajectory is approximately symmetric around this center. **(b)** The vertical velocity zero crossings occur while the pen is at almost fixed vertical levels which correspond to high, normal and low modulation values, yielding altogether 5 quantized levels. The actual pen levels achieved at the vertical velocity zero crossings vary around the quantized values, with approximately normal distribution. Let the indicator, $I_t$ ($I_t \in \{1, \ldots, 5\}$), be the most probable quantized level when the pen is at the position obtained at the $t$'th zero crossing. We need to estimate concurrently the 5 quantized levels $H_1, \ldots, H_5$, their variance $\sigma$ (assumed the same for all levels), and the indicators $I_t$. In this model the *observed data* is the sequence of actual pen levels $L(t)$, while the *complete data* is the sequence of levels and indicators $\{I_t, L(t)\}$. The task of estimating the parameters $\{H_i, \sigma\}$ is performed via *maximum likelihood* estimation from *incomplete data*, commonly done by the *EM* algorithm[1] and described in [9]. The horizontal amplitude is similarly quantized to 3 levels.

After performing slant equalization of the handwriting, namely, orthogonalizing the $x$ and $y$ motions, the velocities $V_x(t), V_y(t)$ become approximately uncorrelated. When $\omega_x \approx \omega_y$, the two velocities are uncorrelated if there is a $\pm 90°$ phase-lag between $V_x$ and $V_y$. There are also locations of total halt in both velocities (no pen movement) which we take as a zero phase lag. Considering the vertical oscillations as a 'master clock', the horizontal oscillations can be viewed as a 'slave clock' whose phase and amplitude vary around the 'master clock'. For English cursive writing, the frequency ratio between the two clocks is limited to the set $\{\frac{1}{2}, 1, 2\}$, thus $V_y$ induces a grid for the possible $V_x$ zero crossings. The phase-lag of the horizontal oscillation is therefore restricted to the values $0°, \pm 90°$ at the zero crossings of $V_y$. The most likely phase-lag trajectory is determined by dynamic programming over the entire grid. At the end of this process the horizontal oscillations are fully determined by the vertical oscillations and the pen trajectory's description greatly simplified.

The variations in the vertical angular velocity for a given writer are small, except in short intervals where the writer hesitates or stops. The only information that should be preserved is the typical vertical angular velocity, denoted by $\omega$. The

normalized discretized equations of motion now become,

$$\begin{cases} \dot{x} = a_i \sin(\omega t + \phi_j) + 1 & a_i \in \{A_1^x, A_2^x, A_3^x\} \ \phi_j \in \{-90°, 0°, 90°\} \\ \dot{y} = b_k \sin(\omega t) & b_k \in \{H_{l_2} - H_{l_1} \mid 1 \le l_1, l_2 \le 5\} \ . \end{cases} \quad (2)$$

We used *analysis by synthesis* technique in order to verify our assumptions and estimation scheme. The final result of the whole process is depicted in Fig. 1, where the original handwriting is plotted together with its reconstruction from the discrete representation.

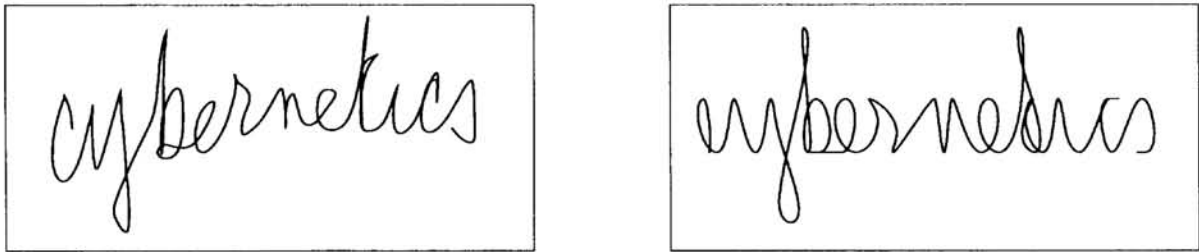

Figure 1: The original and the fully quantized cursive scripts.

## 3  Discrete control sequences

The process described in the previous section results in a *many to one* mapping from the continuous velocity field, $V_x(t), V_y(t)$, to a discrete set of symbols. This set is composed of the cartesian product of the quantized vertical and horizontal amplitudes and the phase-lags between these velocities. We treat this discrete control sequence as a cartesian product time series. Using the value '0' to indicate that the corresponding oscillation continues with the same dynamics, a change in the phase lag can be encoded by setting the code to zero for one dimension, while switching to a new value in the other dimension. A zero in both dimensions indicates no activity. In this way we can model *'pen ups'* intervals and incorporate auxiliary symbols like 'dashes','dots', and 'crosses', that play an important role in resolving disambiguations between letters. These auxiliary are modeled as a separate channel and are ordered according to their $X$ coordinate. We encode the control levels by numbers from **1** to **5**, for the 5 levels of vertical positions. The quantized horizontal amplitudes are coded by 5 values as well: 2 for positive amplitudes (small and large), 2 for negative amplitudes, and one for zero amplitude. Below is an example of our discrete representation for the handwriting depicted in Fig. 1. The upper and lower lines encode the vertical and horizontal oscillations respectively, and the auxiliary channel is omitted. In this example there is only one location where both symbols are '0', indicating a pen-up at the end of the word.

240204204001005002040202204020402424204020500204020402400440240220
104034030410420320401050010502425305010502041032403050033105001000

## 4  Stochastic modeling of the motor control sequences

Existing stochastic modeling methods, such as *Hidden Markov Models* (HMM) [3], suffer from several serious drawbacks. They suffer from the need to 'fix' a-priory the

architecture of the model; they require large amounts of *segmented* training data; and they are very hard to adapt to new data. The stochastic model presented here is an on-line learning algorithm whose important property is its simple adaptability to new examples. We begin with a brief introduction to probabilistic automata, leaving the theoretical issues and some of the more technical details to another place.

A *probabilistic automaton* is a 6-tuple $(Q, \Sigma, \tau, \gamma, q_s, q_e)$, where $Q$ is a finite set of $n$ *states*, $\Sigma$ is an *alphabet* of size $k$, $\tau : Q \times \Sigma \rightarrow Q$ is the state transition function, $\gamma : \Sigma \times Q \rightarrow [0, 1]$ is the transition (output) probability where for every $q \in Q$, $\sum_{\sigma \in \Sigma} \gamma(\sigma|q) = 1$. $q_s \in Q$ is a start state, and $q_e \in Q$ is an end state. A probabilistic automaton is called *acyclic* if it contains no cycles. We denote such automata by PAA. This type of automaton is also known as a *Markov* process with a single source and a single absorbing state. The rest of the states are all *transient* states. Such automata induce non-zero probabilities on a *finite* set of strings. Given an input string $\bar{\sigma} = (\sigma_1, \ldots, \sigma_n)$ if at the of end its 'run' the automaton entered the final state $q_e$, the probability of a string $\bar{\sigma}$ is defined to be, $P(\bar{\sigma}) = \prod_{i=1}^{N} \gamma(\sigma_i|q_{i-1})$ where $q_0 = q_s$, $q_i = \tau(q_{i-1}, \sigma_i)$. On the other hand, if $q_N \neq q_e$ then $P(\bar{\sigma}) = 0$.

The inference of the PAA structure from data can be viewed as a communication problem. Suppose that one wants to transmit an ensemble of strings, all created by the same PAA. If both sides know the structure and probabilities of the PAA then the transmitter can optimally encode the strings by using the PAA transition probabilities. If only the transmitter knows the structure and the receiver has to discover it while receiving new strings, each time a new transition occurs, the transmitter has to send the next state index as well. Since the automaton is acyclic, the possible next states are limited to those which do not form a cycle when the new edge is added to the automaton. Let $k_q^t$ be the number of legal next states from a state $q$ known to the receiver at time $t$. Then the encoding of the next state index requires at least $\log_2(k_q^t + 1)$ bits. The receiver also needs to estimate the state transition probability from the previously received strings. Let $n(\sigma|q)$ be the number of times the symbol $\sigma$ has been observed by the receiver while being in state $q$. Then the transition probability is estimated by Laplace's rule of succession, $\hat{P}(\sigma|q) = \frac{n(\sigma|q)+1}{\sum_{\sigma' \in \Sigma} n(\sigma'|q)+|\Sigma|}$. In sum, if $q$ is the current state and $k_q^t$ the number of possible next states known to the receiver, the number of bits required to encode the next symbol $\sigma$ (assuming optimal coding scheme) is given by: **(a)** if the transition $\tau(q, \sigma)$ has already been observed: $-\log_2(\hat{P}(\sigma|q))$ ; **(b)** if the transition $\tau(q, \sigma)$ has never occurred before: $-\log_2(\hat{P}(\sigma|q)) + \log_2(k_q^t + 1)$.

In training such a model from empirical observations it is necessary to infer the structure of the PAA as well its parameters. We can thus use the above coding scheme to find a minimal description length (MDL) of the data, provided that our model assumption is correct. Since the true PAA is not known to us, we need to imitate the role of the receiver in order to find the optimal coding of a message. This can be done efficiently via dynamic programming for each individual string. After the optimal coding for a single string has been found, the new states are added, the transition probabilities $\hat{P}(\sigma|q)$ are updated and the number of legal next states $k_q$ is recalculated. An example of the learning algorithm is given in Fig. 2, with the estimated probabilities $\hat{P}$, written on the graph edges.

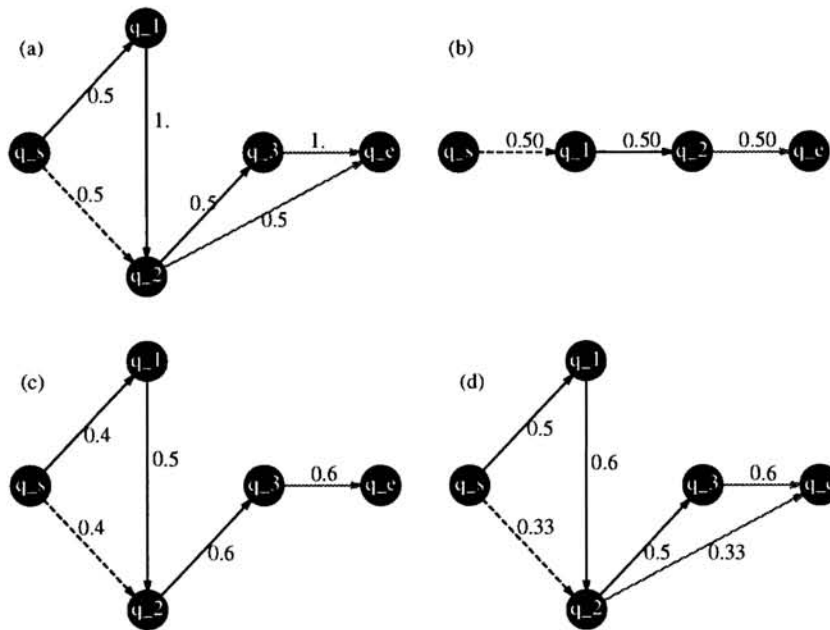

Figure 2: Demonstration of the PAA learning algorithm. Figure (a) shows the original automaton from which the examples were created. Figures (b)-(d) are the intermediate automata built by the algorithm. Edges drawn with bold, dashed, and grey lines correspond to transitions with the symbols '0', '1', and the terminating symbol, respectively.

## 5   Automatic segmentation of cursive scripts

Since the learning algorithm of a PAA is an on-line scheme, only a small number of segmented examples is needed in order to built an initial model. For cursive handwriting we manually collected and segmented about 10 examples, for each lower case cursive letter, and built 26 initial models. At this stage the models are small and do not capture the full variability of the control sequences. Yet this set of initial automata was sufficient to gradually segment cursive scripts into letters and update the models from these segments. Segmented words with high likelihood are fed back into the learning algorithm and the models are further refined. The process is iterated until all the training data is segmented with high likelihood.

The likelihood of new data might not be defined due the incompleteness of the automata, hence the learning algorithm is again applied in order to induce probabilities. Let $P_{i,j}^{S}$ be the probability that a model $S$ (which represents a cursive letter) generates the control symbols $s_i, \ldots, s_{j-1}$ $(j > i)$. The log-likelihood of a proposed segmentation $(i_1, i_2, \ldots, i_{N+1})$ of a word $S_1, S_2, \ldots, S_N$ is,

$$L\left((i_1, \ldots, i_{N+1})|(S_1, \ldots, S_N), (s_1, \ldots, s_L)\right) = \log(\prod_{j=1}^{N} P_{i_j, i_{j+1}}^{S_j}) = \sum_{j=1}^{N} \log(P_{i_j, i_{j+1}}^{S_j}) \ .$$

The segmentation is calculated efficiently by maintaining a layers graph and using dynamic programming to compute recursively the most likely segmentation. Formally, let $ML(n, k)$ be the highest likelihood segmentation of the word up to the

$n$'th control symbol and the $k$'th letter in the word. Then,

$$ML(n,k) = \max_{i_{k-1} \leq i \leq n} \left\{ ML(i, k-1) + \log\left(P_{i,n}^{S_k}\right) \right\}$$

The best segmentation is obtained by tracking the most likely path from $M(N, L)$ back to $M(1, 1)$. The result of such a segmentation is depicted in Fig. 3.

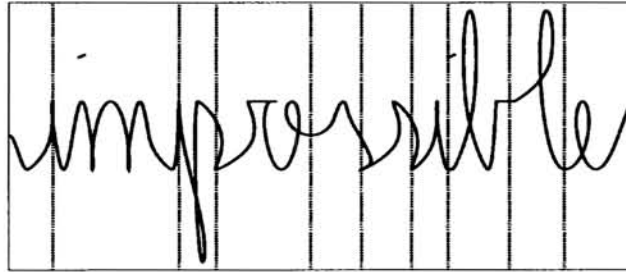

Figure 3: *Temporal* segmentation of the word **impossible**. The segmentation is performed by applying the automata of the letters contained in the word, and finding the Maximum-Likelihood sequence of models via dynamic programming.

## 6  Inducing probabilities for unlabeled words

Using this scheme we automatically segmented a database which contained about 1200 frequent english words, by three different writers. After adding the segmented letters to the training set the resulting automata were general enough, yet very compact. Thus inducing probabilities and recognition of unlabeled data could be performed efficiently. The probability of locating letters in certain locations in new unlabeled words (i.e. words whose transcription is not given) can be evaluated by the automata. These probabilities are calculated by applying the various models on each sub-string of the control sequence, in parallel. Since the automata can accommodate different lengths of observations, the log-likelihood should be divided by the length of the sequence. This normalized log-likelihood is an approximation of the entropy induced by the models, and measures the uncertainty in determining the transcription of a word. The score which measures the uncertainty of the occurrence of a letter $S$ in place $n$ in the a word is, $Score(n|S) = \max_l \frac{1}{l}\log(P_{n,n+l-1}^S)$. The result of applying several automata to a new word is shown in Fig. 4. High probability of a given automaton indicates a beginning of a letter with the corresponding model. The probabilities for the letters **k**, **a**, **e**, **b** are plotted top to bottom. The correspondence between high likelihood points and the relevant locations in the words are shown with dashed lines. These locations occur near the 'true' occurrence of the letter and indicate that these probabilities can be used for recognition and spotting of cursive handwriting. There are other locations where the automata obtain high scores. These correspond to words with high similarity to the model letter and can be resolved by higher level models, similar to techniques used in speech.

## 7  Conclusions and future research

In this paper we present a novel stochastic modeling approach for the analysis, spotting, and recognition of online cursive handwriting. Our scheme is based on a

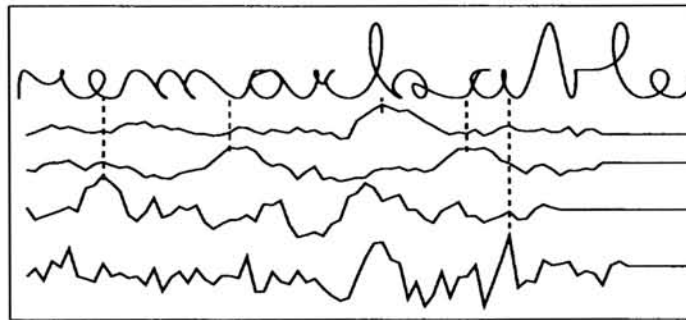

Figure 4: The normalized log-likelihood scores induced by the automata for the letters **k**, **a**, **e**, and **b** (top to bottom). Locations with high score are marked with dashed lines and indicate the relative positions of the letters in the word.

discrete dynamic representation of the handwriting trajectory, followed by training adaptive probabilistic automata for frequent writing sequences. These automata are easy to train and provide simple adaptation mechanism with sufficient power to capture the high variability of cursively written words. Preliminary experiments show that over 90% of the single letters are correctly identified and located, without any additional higher level language model. Methods for higher level statistical language models are also being investigated [6], and will be incorporated into a complete recognition system.

## Acknowledgments

We would like to thank Dana Ron for useful discussions and Lee Giles for providing us with the software for plotting finite state machines. Y.S. would like to thank the Clore foundation for its support.

## References

[1] A. Dempster, N. Laird, and D. Rubin. Maximum likelihood estimation from incomplete data via the EM algorithm. *J. Roy. Statist. Soc.*, 39(B):1–38, 1977.

[2] J.M. Hollerbach. An oscillation theory of handwriting. *Bio. Cyb.*, 39, 1981.

[3] L.R. Rabiner. A tutorial on hidden markov models and selected applications in speech recognition. *Proc. IEEE*, pages 257–286, Feb. 1989.

[4] J. Rissanen. Modeling by shortest data description. *Automatica*, 14, 1978.

[5] J. Rissanen. Stochastic complexity and modeling. *Annals of Stat.*, 14(3), 1986.

[6] D. Ron, Y. Singer, and N. Tishby. The power of amnesia. In *this volume*.

[7] D.E. Rumelhart. Theory to practice: a case study - recognizing cursive handwriting. In *Proc. of 1992 NEC Conf. on Computation and Cognition*.

[8] Y. Singer and N. Tishby. Dynamical encoding of cursive handwriting. In *IEEE Conference on Computer Vision and Pattern Recognition*, 1993.

[9] Y. Singer and N. Tishby. Dynamical encoding of cursive handwriting. Technical Report CS93-4, The Hebrew University of Jerusalem, 1993.
